# Hierarchical Bayesian Inference in Networks of Spiking Neurons

**Rajesh P. N. Rao**
Department of Computer Science and Engineering
University of Washington, Seattle, WA 98195
*rao@cs.washington.edu*

## Abstract

There is growing evidence from psychophysical and neurophysiological studies that the brain utilizes Bayesian principles for inference and decision making. An important open question is how Bayesian inference for arbitrary graphical models can be implemented in networks of spiking neurons. In this paper, we show that recurrent networks of noisy integrate-and-fire neurons can perform approximate Bayesian inference for dynamic and hierarchical graphical models. The membrane potential dynamics of neurons is used to implement belief propagation in the log domain. The spiking probability of a neuron is shown to approximate the posterior probability of the preferred state encoded by the neuron, given past inputs. We illustrate the model using two examples: (1) a motion detection network in which the spiking probability of a direction-selective neuron becomes proportional to the posterior probability of motion in a preferred direction, and (2) a two-level hierarchical network that produces attentional effects similar to those observed in visual cortical areas V2 and V4. The hierarchical model offers a new Bayesian interpretation of attentional modulation in V2 and V4.

## 1 Introduction

A wide range of psychophysical results have recently been successfully explained using Bayesian models [7, 8, 16, 19]. These models have been able to account for human responses in tasks ranging from 3D shape perception to visuomotor control. Simultaneously, there is accumulating evidence from human and monkey experiments that Bayesian mechanisms are at work during visual decision making [2, 5]. The versatility of Bayesian models stems from their ability to combine prior knowledge with sensory evidence in a rigorous manner: Bayes rule prescribes how prior probabilities and stimulus likelihoods should be combined, allowing the responses of subjects or neural responses to be interpreted in terms of the resulting posterior distributions.

An important question that has only recently received attention is how networks of cortical neurons can implement algorithms for Bayesian inference. One powerful approach has been to build on the known properties of population coding models that represent information using a set of neural tuning curves or kernel functions [1, 20]. Several proposals have been made regarding how a probability distribution could be encoded using population codes ([3, 18]; see [14] for an excellent review). However, the problem of implementing general inference algorithms for arbitrary graphical models using population codes remains unresolved (some encouraging initial results are reported in Zemel *et al.*, this volume). An

alternate approach advocates performing Bayesian inference in the log domain such that multiplication of probabilities is turned into addition and division to subtraction, the latter operations being easier to implement in standard neuron models [2, 5, 15] (see also the papers by Deneve and by Yu and Dayan in this volume). For example, a neural implementation of approximate Bayesian inference for a hidden Markov model was investigated in [15]. The question of how such an approach could be generalized to spiking neurons and arbitrary graphical models remained open.

In this paper, we propose a method for implementing Bayesian belief propagation in networks of spiking neurons. We show that recurrent networks of noisy integrate-and-fire neurons can perform approximate Bayesian inference for dynamic and hierarchical graphical models. In the model, the dynamics of the membrane potential is used to implement on-line belief propagation in the log domain [15]. A neuron's spiking probability is shown to approximate the posterior probability of the preferred state encoded by the neuron, given past inputs. We first show that for a visual motion detection task, the spiking probability of a direction-selective neuron becomes proportional to the posterior probability of motion in the neuron's preferred direction. We then show that in a two-level network, hierarchical Bayesian inference [9] produces responses that mimic the attentional effects seen in visual cortical areas V2 and V4.

## 2 Modeling Networks of Noisy Integrate-and-Fire Neurons

### 2.1 Integrate-and-Fire Model of Spiking Neurons

We begin with a recurrently-connected network of integrate-and-fire (IF) neurons receiving feedforward inputs denoted by the vector $\mathbf{I}$. The membrane potential of neuron $i$ changes according to:

$$\tau \frac{dv_i}{dt} = -v_i + \sum_j w_{ij} I_j + \sum_j u_{ij} v_j' \tag{1}$$

where $\tau$ is the membrane time constant, $I_j$ denotes the synaptic current due to input neuron $j$, $w_{ij}$ represents the strength of the synapse from input $j$ to recurrent neuron $i$, $v_j'$ denotes the synaptic current due to recurrent neuron $j$, and $u_{ij}$ represents the corresponding synaptic strength. If $v_i$ crosses a threshold $T$, the neuron fires a spike and $v_i$ is reset to the potential $v_{reset}$. Equation 1 can be rewritten in discrete form as:

$$v_i(t+1) = v_i(t) + \epsilon(-v_i(t) + \sum_j w_{ij} I_j(t)) + \sum_j u_{ij} v_j'(t) \tag{2}$$

$$\text{i.e.} \quad v_i(t+1) = \epsilon \sum_j w_{ij} I_j(t) + \sum_j u_{ij}' v_j'(t) \tag{3}$$

where $\epsilon$ is the integration rate, $u_{ii}' = 1 + \epsilon(u_{ii} - 1)$ and for $i \neq j$, $u_{ij}' = \epsilon u_{ij}$.

A more general integrate-and-fire model that takes into account some of the effects of nonlinear filtering in dendrites can be obtained by generalizing Equation 3 as follows:

$$v_i(t+1) = f\left(\sum_j w_{ij} I_j(t)\right) + g\left(\sum_j u_{ij}' v_j'(t)\right) \tag{4}$$

where $f$ and $g$ model potentially different dendritic filtering functions for feedforward and recurrent inputs.

### 2.2 Stochastic Spiking in Noisy IF Neurons

To model the effects of background inputs and the random openings of membrane channels, one can add a Gaussian white noise term to the right hand side of Equations 3 and 4. This makes the spiking of neurons in the recurrent network stochastic. Plesser and Gerstner [13] and Gerstner [4] have shown that under reasonable assumptions, the probability of spiking

in such noisy neurons can be approximated by an "escape function" (or hazard function) that depends only on the distance between the (noise-free) membrane potential $v_i$ and the threshold $T$. Several different escape functions were studied. Of particular interest to the present paper is the following exponential function for spiking probability suggested in [4] for noisy integrate-and-fire networks:

$$P(\text{neuron } i \text{ spikes at time } t) = ke^{(v_i(t)-T)/c} \qquad (5)$$

where $k$ and $c$ are arbitrary constants. We used a model that combines Equations 4 and 5 to generate spikes, with an absolute refractory period of 1 time step.

## 3 Bayesian Inference using Spiking Neurons

### 3.1 Inference in a Single-Level Model

We first consider on-line belief propagation in a single-level dynamic graphical model and show how it can be implemented in spiking networks. The graphical model is shown in Figure 1A and corresponds to a classical hidden Markov model. Let $\theta(t)$ represent the hidden state of a Markov model at time $t$ with transition probabilities given by $P(\theta(t) = \theta_i | \theta(t-1) = \theta_j) = P(\theta_i^t | \theta_j^{t-1})$ for $i, j = 1 \ldots N$. Let $\mathbf{I}(t)$ be the observable output governed by the probabilities $P(\mathbf{I}(t)|\theta(t))$. Then, the forward component of the belief propagation algorithm [12] prescribes the following "message" for state $i$ from time step $t$ to $t+1$:

$$m_i^{t,t+1} = P(\mathbf{I}(t)|\theta_i^t) \sum_j P(\theta_i^t | \theta_j^{t-1}) m_j^{t-1,t} \qquad (6)$$

If $m_i^{0,1} = P(\theta_i)$ (the prior distribution over states), then it is easy to show using Bayes rule that $m_i^{t,t+1} = P(\theta_i^t, \mathbf{I}(t), \ldots, \mathbf{I}(1))$. If the probabilities are normalized at each update step:

$$m_i^{t,t+1} = P(\mathbf{I}(t)|\theta_i^t) \sum_j P(\theta_i^t | \theta_j^{t-1}) m_j^{t-1,t} / n^{t-1,t} \qquad (7)$$

where $n^{t-1,t} = \sum_j m_j^{t-1,t}$, then the message becomes equal to the posterior probability of the state and current input, given all past inputs:

$$m_i^{t,t+1} = P(\theta_i^t, \mathbf{I}(t)|\mathbf{I}(t-1), \ldots, \mathbf{I}(1)) \qquad (8)$$

### 3.2 Neural Implementation of the Inference Algorithm

By comparing the membrane potential equation (Eq. 4) with the on-line belief propagation equation (Eq. 7), it is clear that the first equation can implement the second if *belief propagation is performed in the log domain* [15], i.e., if:

$$v_i(t+1) \quad \propto \quad \log m_i^{t,t+1} \qquad (9)$$

$$f\left(\sum_j w_{ij} I_j(t)\right) \quad = \quad \log P(\mathbf{I}(t)|\theta_i^t) \qquad (10)$$

$$g\left(\sum_j u_{ij}' v_j'(t)\right) \quad = \quad \log\left(\sum_j P(\theta_i^t | \theta_j^{t-1}) m_j^{t-1,t} / n^{t-1,t}\right) \qquad (11)$$

In this model, the dendritic filtering functions $f$ and $g$ approximate the logarithm function[1], the synaptic currents $I_j(t)$ and $v_j'(t)$ are approximated by the corresponding instantaneous firing rates, and the recurrent synaptic weights $u_{ij}'$ encode the transition probabilities $P(\theta_i^t | \theta_j^{t-1})$. Normalization by $n^{t-1,t}$ is implemented by subtracting $\log n^{t-1,t}$ using inhibition.

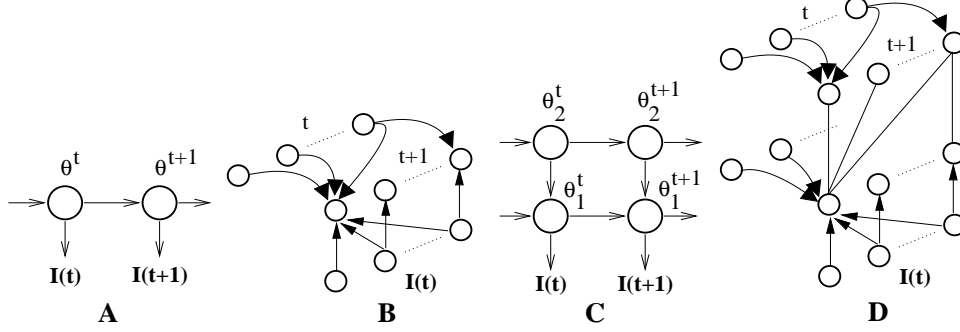

Figure 1: **Graphical Models and their Neural Implementation**. (A) Single-level dynamic graphical model. Each circle represents a node denoting the state variable $\theta^t$ which can take on values $\theta_1, \ldots, \theta_N$. (B) Recurrent network for implementing on-line belief propagation for the graphical model in (A). Each circle represents a neuron encoding a state $\theta_i$. Arrows represent synaptic connections. The probability distribution over state values at each time step is represented by the entire population. (C) Two-level dynamic graphical model. (D) Two-level network for implementing on-line belief propagation for the graphical model in (C). Arrows represent synaptic connections in the direction pointed by the arrow heads. Lines without arrow heads represent bidirectional connections.

Finally, since the membrane potential $v_i(t+1)$ is assumed to be proportional to $\log m_i^{t,t+1}$ (Equation 9), we have:

$$v_i(t+1) = c \log m_i^{t,t+1} + T \tag{12}$$

for some constants $c$ and $T$. For noisy integrate-and-fire neurons, we can use Equation 5 to calculate the probability of spiking for each neuron $i$ as:

$$P(\text{neuron } i \text{ spikes at time } t+1) \quad \propto \quad e^{(v_i(t+1)-T)/c} \tag{13}$$

$$= \quad e^{\log m_i^{t,t+1}} = m_i^{t,t+1} \tag{14}$$

Thus, the probability of spiking (or equivalently, the instantaneous firing rate) for neuron $i$ in the recurrent network is directly proportional to the posterior probability of the neuron's preferred state and the current input, given all past inputs. Figure 1B illustrates the single-level recurrent network model that implements the on-line belief propagation equation 7.

### 3.3 Hierarchical Inference

The model described above can be extended to perform on-line belief propagation and inference for arbitrary graphical models. As an example, we describe the implementation for the two-level hierarchical graphical model in Figure 1C.

As in the case of the 1-level dynamic model, we define the following "messages" within a particular level and between levels: $m_{1,i}^{t,t+1}$ (message from state $i$ to other states at level 1 from time step $t$ to $t+1$), $m_{1\to2,i}^t$ ("feedforward" message from state $i$ at level 1 sent to level 2 at time $t$), $m_{2,i}^{t,t+1}$ (message from state $i$ to other states at level 2 from time step $t$ to $t+1$), and $m_{2\to1,i}^t$ ("feedback" message from state $i$ at level 2 sent to level 1 at time $t$). Each of these messages can be calculated based on an on-line version of loopy belief propagation [11] for the multiply connected two-level graphical model in Figure 1C:

$$m_{1\to2,i}^t = \sum_j \sum_k P(\theta_{1,k}^t|\theta_{2,i}^t, \theta_{1,j}^{t-1})m_{1,j}^{t-1,t}P(\mathbf{I}(t)|\theta_{1,k}^t) \tag{15}$$

$$m_{2\to1,i}^t = \sum_j P(\theta_{2,i}^t|\theta_{2,j}^{t-1})m_{2,j}^{t-1,t} \tag{16}$$

$$m_{1,i}^{t,t+1} \quad = \quad P(\mathbf{I}(t)|\theta_{1,i}^t)\Big(\sum_j \sum_k P(\theta_{1,i}^t|\theta_{2,j}^t,\theta_{1,k}^{t-1})m_{2\to 1,j}^t m_{1,k}^{t-1,t}\Big) \qquad (17)$$

$$m_{2,i}^{t,t+1} \quad = \quad m_{1\to 2,i}^t\Big(\sum_j P(\theta_{2,i}^t|\theta_{2,j}^{t-1})m_{2,j}^{t-1,t}\Big) \qquad (18)$$

Note the similarity between the last equation and the equation for the single-level model (Equation 6). The equations above can be implemented in a 2-level hierarchical recurrent network of integrate-and-fire neurons in a manner similar to the 1-level case. We assume that neuron $i$ in level 1 encodes $\theta_{1,i}$ as its preferred state while neuron $i$ in level 2 encodes $\theta_{2,i}$. We also assume specific feedforward and feedback neurons for computing and conveying $m_{1\to 2,i}^t$ and $m_{2\to 1,i}^t$ respectively.

Taking the logarithm of both sides of Equations 17 and 18, we obtain equations that can be computed using the membrane potential dynamics of integrate-and-fire neurons (Equation 4). Figure 1D illustrates the corresponding two-level hierarchical network. A modification needed to accommodate Equation 17 is to allow bilinear interactions between synaptic inputs, which changes Equation 4 to:

$$v_i(t+1) = f\Big(\sum_j w_{ij}I_j(t)\Big) + g\Big(\sum_j \sum_k u'_{ijk}v'_j(t)x_k(t)\Big) \qquad (19)$$

Multiplicative interactions between synaptic inputs have previously been suggested by several authors (e.g., [10]) and potential implementations based on active dendritic interactions have been explored. The model suggested here utilizes these multiplicative interactions within dendritic branches, in addition to a possible logarithmic transform of the signal before it sums with other signals at the soma. Such a model is comparable to recent models of dendritic computation (see [6] for more details).

## 4   Results

### 4.1   Single-Level Network: Probabilistic Motion Detection and Direction Selectivity

We first tested the model in a 1D visual motion detection task [15]. A single-level recurrent network of 30 neurons was used (see Figure 1B). Figure 2A shows the feedforward weights for neurons $1, \ldots, 15$: these were recurrently connected to encode transition probabilities biased for rightward motion as shown in Figure 2B. Feedforward weights for neurons $16, \ldots, 30$ were identical to Figure 2A but their recurrent connections encoded transition probabilities for leftward motion (see Figure 2B). As seen in Figure 2C, neurons in the network exhibited direction selectivity. Furthermore, the spiking probability of neurons reflected the posterior probabilities over time of motion direction at a given location (Figure 2D), suggesting a probabilistic interpretation of direction selective spiking responses in visual cortical areas such as V1 and MT.

### 4.2   Two-Level Network: Spatial Attention as Hierarchical Bayesian Inference

We tested the two-level network implementation (Figure 1D) of hierarchical Bayesian inference using a simple attention task previously used in primate studies [17]. In an input image, a vertical or horizontal bar could occur either on the left side, right side, or both sides (see Figure 3). The corresponding 2-level generative model consisted of two states at level 2 (left or right side) and four states at level 1: vertical left, horizontal left, vertical right, horizontal right. Each of these states was encoded by a neuron at the respective level. The feedforward connections at level 1 were chosen to be vertically or horizontally oriented Gabor filters localized to the left or right side of the image. Since the experiment used static images, the recurrent connections at each level implemented transition probabilities close to 1 for the same state and small random values for other states. The transition probabilities $P(\theta_{1,k}^t|\theta_{2,i}^t,\theta_{1,j}^{t-1})$ were chosen such that for $\theta_2^t =$ left side, the transition probabilities for

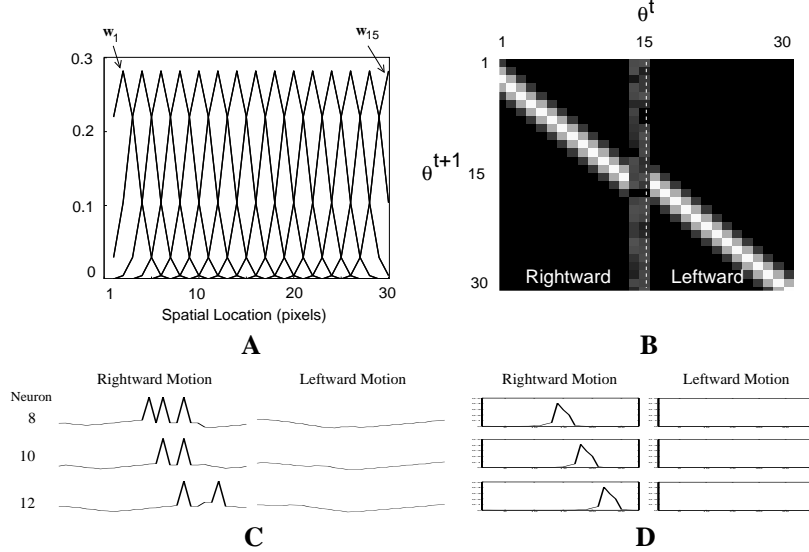

**A**                **B**

**C**                **D**

Figure 2: **Responses from the Single-Level Motion Detection Network**. (A) Feedforward weights for neurons $1, \ldots, 15$ (rightward motion selective neurons). Feedforward weights for neurons $16, \ldots, 30$ (leftward motion selective) are identical. (B) Recurrent weights encoding the transition probabilities $P(\theta_i^{t+1}|\theta_j^t)$ for $i, j = 1, \ldots, 30$. Probability values are proportional to pixel brightness. (C) Spiking responses of three of the first 15 neurons in the recurrent network (neurons 8, 10, and 12). As is evident, these neurons have become selective for rightward motion as a consequence of the recurrent connections specified in (B). (D) Posterior probabilities over time of motion direction (at a given location) encoded by the three neurons for rightward and leftward motion.

states $\theta_1^t$ coding for the right side were set to values close to zero (and vice versa, for $\theta_2^t =$ right side). As shown in Figure 3, the response of a neuron at level 1 that, for example, prefers a vertical edge on the right mimics the response of a V4 neuron with and without attention (see figure caption for more details). The initial setting of the priors at level 2 is the crucial determinant of attentional modulation in level 1 neurons, suggesting that feedback from higher cortical areas may convey task-specific priors that are integrated into V4 responses.

## 5    Discussion and Conclusions

We have shown that recurrent networks of noisy integrate-and-fire neurons can perform approximate Bayesian inference for single- and multi-level dynamic graphical models. The model suggests a new interpretation of the spiking probability of a neuron in terms of the posterior probability of the preferred state encoded by the neuron, given past inputs. We illustrated the model using two problems: inference of motion direction in a single-level network and hierarchical inference of object identity at an attended visual location in a two-level network. In the first case, neurons generated direction-selective spikes encoding the probability of motion in a particular direction. In the second case, attentional effects similar to those observed in primate cortical areas V2 and V4 emerged as a result of imposing appropriate priors at the highest level.

The results obtained thus far are encouraging but several important questions remain. How does the approach scale to more realistic graphical models? The two-level model explored in this paper assumed stationary objects, resulting in simplified dynamics for the two levels in our recurrent network. Experiments are currently underway to test the robustness of the proposed model when richer classes of dynamics are introduced at the different levels. An-

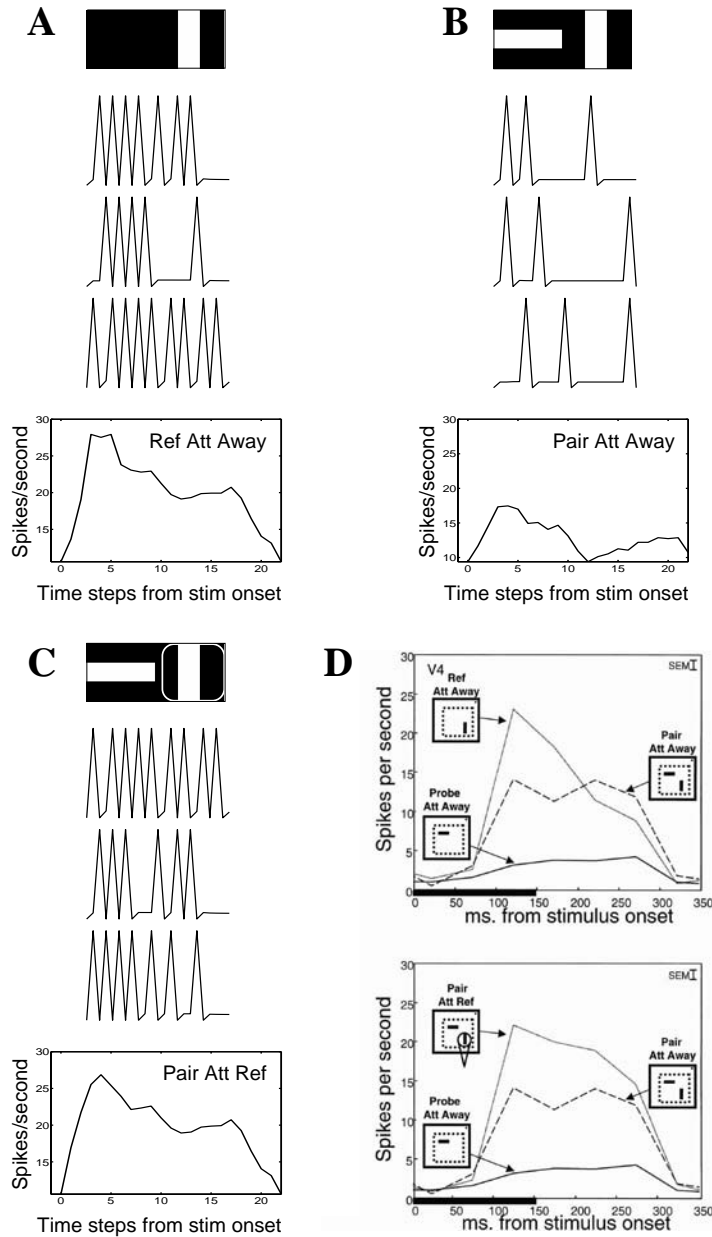

Figure 3: **Responses from the Two-Level Hierarchical Network**. (A) Top panel: Input image (lasting the first 15 time steps) containing a vertical bar ("Reference") on the right side. Each input was convolved with a retinal spatiotemporal filter. Middle: Three sample spike trains from the 1st level neuron whose preferred stimulus was a vertical bar on the right side. Bottom: Posterior probability of a vertical bar (= spiking probability or instantaneous firing rate of the neuron) plotted over time. (B) Top panel: An input containing two stimuli ("Pair"). Below: Sample spike trains and posterior probability for the same neuron as in (A). (C) When "attention" is focused on the right side (depicted by the white oval) by initializing the prior probability encoded by the 2nd level right-coding neuron at a higher value than the left-coding neuron, the firing rate for the 1st level neuron in (A) increases to a level comparable to that in (A). (D) Responses from a neuron in primate area V4 without attention (top panel, Ref Att Away and Pair Att Away; compare with (A) and (B)) and with attention (bottom panel, Pair Att Ref; compare with (C)) (from [17]). Similar responses are seen in V2 [17].

other open question is how active dendritic processes could support probabilistic integration of messages from local, lower-level, and higher-level neurons, as suggested in Section 3. We intend to investigate this question using biophysical (compartmental) models of cortical neurons. Finally, how can the feedforward, feedback, and recurrent synaptic weights in the networks be learned directly from input data? We hope to investigate this question using biologically-plausible approximations to the expectation-maximization (EM) algorithm.

**Acknowledgments**. This research was supported by grants from ONR, NSF, and the Packard Foundation. I am grateful to Wolfram Gerstner, Michael Shadlen, Aaron Shon, Eero Simoncelli, and Yair Weiss for discussions on topics related to this paper.

## Footnotes

[1] An alternative approach, which was also found to yield satisfactory results, is to approximate the log-sum with a linear weighted sum [15], the weights being chosen to minimize the approximation error.

# References

[1] C. H. Anderson and D. C. Van Essen. Neurobiological computational systems. In *Computational Intelligence: Imitating Life*, pages 213–222. New York, NY: IEEE Press, 1994.

[2] R. H. S. Carpenter and M. L. L. Williams. Neural computation of log likelihood in control of saccadic eye movements. *Nature*, 377:59–62, 1995.

[3] S. Deneve and A. Pouget. Bayesian estimation by interconnected neural networks (abstract no. 237.11). *Society for Neuroscience Abstracts*, 27, 2001.

[4] W. Gerstner. Population dynamics of spiking neurons: Fast transients, asynchronous states, and locking. *Neural Computation*, 12(1):43–89, 2000.

[5] J. I. Gold and M. N. Shadlen. Neural computations that underlie decisions about sensory stimuli. *Trends in Cognitive Sciences*, 5(1):10–16, 2001.

[6] M. Hausser and B. Mel. Dendrites: bug or feature? *Current Opinion in Neurobiology*, 13:372–383, 2003.

[7] D. C. Knill and W. Richards. *Perception as Bayesian Inference*. Cambridge, UK: Cambridge University Press, 1996.

[8] K. P. Körding and D. Wolpert. Bayesian integration in sensorimotor learning. *Nature*, 427:244–247, 2004.

[9] T. S. Lee and D. Mumford. Hierarchical Bayesian inference in the visual cortex. *Journal of the Optical Society of America A*, 20(7):1434–1448, 2003.

[10] B. W. Mel. NMDA-based pattern discrimination in a modeled cortical neuron. *Neural Computation*, 4(4):502–517, 1992.

[11] K. Murphy, Y. Weiss, and M. Jordan. Loopy belief propagation for approximate inference: An empirical study. In *Proceedings of UAI (Uncertainty in AI)*, pages 467–475. 1999.

[12] J. Pearl. *Probabilistic Reasoning in Intelligent Systems: Networks of Plausible Inference*. Morgan Kaufmann, San Mateo, CA, 1988.

[13] H. E. Plesser and W. Gerstner. Noise in integrate-and-fi re neurons: From stochastic input to escape rates. *Neural Computation*, 12(2):367–384, 2000.

[14] A. Pouget, P. Dayan, and R. S. Zemel. Inference and computation with population codes. *Annual Review of Neuroscience*, 26:381–410, 2003.

[15] R. P. N. Rao. Bayesian computation in recurrent neural circuits. *Neural Computation*, 16(1):1–38, 2004.

[16] R. P. N. Rao, B. A. Olshausen, and M. S. Lewicki. *Probabilistic Models of the Brain: Perception and Neural Function*. Cambridge, MA: MIT Press, 2002.

[17] J. H. Reynolds, L. Chelazzi, and R. Desimone. Competitive mechanisms subserve attention in macaque areas V2 and V4. *Journal of Neuroscience*, 19:1736–1753, 1999.

[18] M. Sahani and P. Dayan. Doubly distributional population codes: Simultaneous representation of uncertainty and multiplicity. *Neural Computation*, 15:2255–2279, 2003.

[19] Y. Weiss, E. P. Simoncelli, and E. H. Adelson. Motion illusions as optimal percepts. *Nature Neuroscience*, 5(6):598–604, 2002.

[20] R. S. Zemel, P. Dayan, and A. Pouget. Probabilistic interpretation of population codes. *Neural Computation*, 10(2):403–430, 1998.
